# Face Detection — Efficient and Rank Deficient

**Wolf Kienzle, Gökhan Bakır, Matthias Franz and Bernhard Schölkopf**
Max-Planck-Institute for Biological Cybernetics
Spemannstr. 38, D-72076 Tübingen, Germany
{kienzle, gb, mof, bs}@tuebingen.mpg.de

## Abstract

This paper proposes a method for computing fast approximations to support vector decision functions in the field of object detection. In the present approach we are building on an existing algorithm where the set of support vectors is replaced by a smaller, so-called reduced set of synthesized input space points. In contrast to the existing method that finds the reduced set via unconstrained optimization, we impose a structural constraint on the synthetic points such that the resulting approximations can be evaluated via separable filters. For applications that require scanning large images, this decreases the computational complexity by a significant amount. Experimental results show that in face detection, rank deficient approximations are 4 to 6 times faster than unconstrained reduced set systems.

## 1 Introduction

It has been shown that support vector machines (SVMs) provide state-of-the-art accuracies in object detection. In time-critical applications, however, they are of limited use due to their computationally expensive decision functions. In particular, the time complexity of an SVM classification operation is characterized by two parameters. First, it is linear in the number of support vectors (SVs). Second, it scales with the number of operations needed for computing the similarity between an SV and the input, i.e. the complexity of the kernel function. When classifying image patches of size $h \times w$ using plain gray value features, the decision function requires an $h \cdot w$ dimensional dot product for each SV. As the patch size increases, these computations become extremely expensive. As an example, the evaluation of a single $20 \times 20$ patch on a $320 \times 240$ image at 25 frames per second already requires 660 million operations per second.

In the past, research towards speeding up kernel expansions has focused exclusively on the first issue, i.e. on how to reduce the number of expansion points (SVs) [1, 2]. In [2], Burges introduced a method that, for a given SVM, creates a set of so-called reduced set vectors (RSVs) that approximate the decision function. This approach has been successfully applied in the image classification domain — speedups on the order of 10 to 30 have been reported [2, 3, 4] while the full accuracy was retained. Additionally, for strongly unbalanced classification problems such as face detection, the average number of RSV evaluations can be further reduced using cascaded classifiers [5, 6, 7]. Unfortunately, the above example illustrates that even with as few as three RSVs on average (as in [5]), such systems are not competitive for time-critical applications.

The present work focuses on the second issue, i.e. the high computational cost of the kernel evaluations. While this could be remedied by switching to a sparser image representation (e.g. a wavelet basis), one could argue that in connection with SVMs, not only are plain gray values straightforward to use, but they have shown to outperform Haar wavelets and gradients in the face detection domain [8]. Alternatively, in [9], the authors suggest to compute the costly correlations in the frequency domain. In this paper, we develop a method that combines the simplicity of gray value correlations with the speed advantage of more sophisticated image representations. To this end, we borrow an idea from image processing: by constraining the RSVs to have a special structure, they can be evaluated via separable convolutions. This works for most standard kernels (e.g. linear, polynomial, Gaussian and sigmoid) and decreases the average computational complexity of the RSV evaluations from $O(h \cdot w)$ to $O(r \cdot (h + w))$, where $r$ is a small number that allows the user to balance between speed and accuracy. To evaluate our approach, we examine the performance of these approximations on the MIT+CMU face detection database (used in [10, 8, 5, 6]).

## 2 Burges' method for reduced set approximations

The present section briefly describes Burges' reduced set method [2] on which our work is based. For reasons that will become clear below, $h \times w$ image patches are written as $h \times w$ matrices (denoted by bold capital letters) whose entries are the respective pixel intensities. In this paper, we refer to this as the image-matrix notation.

Assume that an SVM has been successfully trained on the problem at hand. Let $\{\mathbf{X}_1, \ldots \mathbf{X}_m\}$ denote the set of SVs, $\{\alpha_1, \ldots \alpha_m\}$ the corresponding coefficients, $k(\cdot, \cdot)$ the kernel function and $b$ the bias of the SVM solution. The decision rule for a test pattern $\mathbf{X}$ reads

$$f(\mathbf{X}) = \text{sgn}\left(\sum_{i=1}^{m} y_i \alpha_i k(\mathbf{X}_i, \mathbf{X}) + b\right). \tag{1}$$

In SVMs, the decision surface induced by $f$ corresponds to a hyperplane in the reproducing kernel Hilbert space (RKHS) associated with $k$. The corresponding normal

$$\Psi = \sum_{i=1}^{m} y_i \alpha_i k(\mathbf{X}_i, \cdot) \tag{2}$$

can be approximated using a smaller, so-called reduced set (RS) $\{\mathbf{Z}_1, \ldots \mathbf{Z}_{m'}\}$ of size $m' < m$, i.e. an approximation to $\Psi$ of the form

$$\Psi' = \sum_{i=1}^{m'} \beta_i k(\mathbf{Z}_i, \cdot). \tag{3}$$

This speeds up the decision process by a factor of $m/m'$. To find such $\Psi'$, we fix a desired set size $m'$ and solve

$$\min \|\Psi - \Psi'\|_{\text{RKHS}}^2 \tag{4}$$

for $\beta_i$ and $\mathbf{Z}_i$. Here, $\|\cdot\|_{\text{RKHS}}$ denotes the Euclidian norm in the RKHS. The resulting RS decision function $f'$ is then given by

$$f'(\mathbf{X}) = \text{sgn}\left(\sum_{i=1}^{m'} \beta_i k(\mathbf{Z}_i, \mathbf{X}) + b\right). \tag{5}$$

In practice, $\beta_i, \mathbf{Z}_i$ are found using a gradient based optimization technique. Details can be found in [2].

# 3 From separable filters to rank deficient reduced sets

We now describe the concept of separable filters in image processing and show how this idea extends to a broader class of linear filters and to a special class of nonlinear filters, namely those used by SVM decision functions. Using the image-matrix notation, it will become clear that the separability property boils down to a matrix rank constraint.

## 3.1 Linear separable filters

Applying a linear filter to an image amounts to a two-dimensional convolution of the image with the impulse response of the filter. In particular, if $\mathbf{I}$ is the input image, $\mathbf{H}$ the impulse response, i.e. the filter mask, and $\mathbf{J}$ the output image, then

$$\mathbf{J} = \mathbf{I} * \mathbf{H}. \tag{6}$$

If $\mathbf{H}$ has size $h \times w$, the convolution requires $O(h \cdot w)$ operations for each output pixel. However, in special cases where $\mathbf{H}$ can be decomposed into two column vectors $\mathbf{a}$ and $\mathbf{b}$, such that

$$\mathbf{H} = \mathbf{a}\mathbf{b}^\top \tag{7}$$

holds, we can rewrite (6) as

$$\mathbf{J} = [\mathbf{I} * \mathbf{a}] * \mathbf{b}^\top, \tag{8}$$

since the convolution is associative and in this case, $\mathbf{a}\mathbf{b}^\top = \mathbf{a} * \mathbf{b}^\top$. This splits the original problem (6) into two convolution operations with masks of size $h \times 1$ and $1 \times w$, respectively. As a result, if a linear filter is separable in the sense of equation (7), the computational complexity of the filtering operation can be reduced from $O(h \cdot w)$ to $O(h + w)$ per pixel by computing (8) instead of (6).

## 3.2 Linear rank deficient filters

In view of (7) being equivalent to $\mathrm{rank}(\mathbf{H}) \leq 1$, we now generalize the above concept to linear filters with low rank impulse responses. Consider the singular value decomposition (SVD) of the $h \times w$ matrix $\mathbf{H}$,

$$\mathbf{H} = \mathbf{U}\mathbf{S}\mathbf{V}^\top, \tag{9}$$

and recall that $\mathbf{U}$ and $\mathbf{V}$ are orthogonal matrices of size $h \times h$ and $w \times w$, respectively, whereas $\mathbf{S}$ is diagonal (the diagonal entries are the singular values) and has size $h \times w$. Now let $r = \mathrm{rank}(\mathbf{H})$. Due to $\mathrm{rank}(\mathbf{S}) = \mathrm{rank}(\mathbf{H})$, we may write $\mathbf{H}$ as a sum of $r$ rank one matrices

$$\mathbf{H} = \sum_{i=1}^{r} s_i \mathbf{u}_i \mathbf{v}_i^\top \tag{10}$$

where $s_i$ denotes the $i$th singular value of $\mathbf{H}$ and $\mathbf{u}_i$, $\mathbf{v}_i$ are the $i$ths columns of $\mathbf{U}$ and $\mathbf{V}$ (i.e. the $i$th singular vectors of the matrix $\mathbf{H}$), respectively. As a result, the corresponding linear filter can be evaluated (analogously to (8)) as the weighted sum of $r$ separable convolutions

$$\mathbf{J} = \sum_{i=1}^{r} s_i [\mathbf{I} * \mathbf{u}_i] * \mathbf{v}_i^\top \tag{11}$$

and the computational complexity drops from $O(h \times w)$ to $O(r \cdot (h + w))$ per output pixel. Not surprisingly, the speed benefit depends on $r$, which can be seen to measure the structural complexity[1] of $\mathbf{H}$. For square matrices ($w = h$) for instance, (11) does not give any speedup compared to (6) if $r > w/2$.

### 3.3 Nonlinear rank deficient filters and reduced sets

Due to the fact that in 2D, correlation is identical with convolution if the filter mask is rotated by 180 degrees (and vice versa), we can apply the above idea to any image filter $f(\mathbf{X}) = g(c(\mathbf{H}, \mathbf{X}))$ where $g$ is an arbitrary nonlinear function and $c(\mathbf{H}, \mathbf{X})$ denotes the correlation between images patches $\mathbf{X}$ and $\mathbf{H}$ (both of size $h \times w$). In SVMs this amounts to using a kernel of the form

$$k(\mathbf{H}, \mathbf{X}) = g(c(\mathbf{H}, \mathbf{X})). \tag{12}$$

If $\mathbf{H}$ has rank $r$, we may split the kernel evaluation into $r$ separable correlations plus a scalar nonlinearity. As a result, if the RSVs in a kernel expansion such as (5) satisfy this constraint, the average computational complexity decreases from $O(m' \cdot h \cdot w)$ to $O(m' \cdot r \cdot (h + w))$ per output pixel. This concept works for many off-the-shelf kernels used in SVMs. While linear, polynomial and sigmoid kernels are defined as functions of input space dot products and therefore immediately satisfy equation (12), the idea applies to kernels based on the Euclidian distance as well. For instance, the Gaussian kernel reads

$$k(\mathbf{H}, \mathbf{X}) = \exp(\gamma(c(\mathbf{X}, \mathbf{X}) - 2c(\mathbf{H}, \mathbf{X}) + c(\mathbf{H}, \mathbf{H}))). \tag{13}$$

Here, the middle term is the correlation which we are going to evaluate via separable filters. The first term is independent of the SVs — it can be efficiently pre-computed and stored in a separate image. The last term is merely a constant scalar independent of the image data. Finally, note that these kernels are usually defined on vectors. Nevertheless, we can use our image-matrix notation due to the fact that the squared Euclidian distance between two vectors of gray values $\mathbf{x}$ and $\mathbf{z}$ may be written as

$$\|\mathbf{x} - \mathbf{z}\|^2 = \|\mathbf{X} - \mathbf{Z}\|_F^2, \tag{14}$$

whereas the dot product amounts to

$$\mathbf{x}^\top \mathbf{z} = \frac{1}{2} \left( \|\mathbf{X}\|_F^2 + \|\mathbf{Z}\|_F^2 - \|\mathbf{X} - \mathbf{Z}\|_F^2 \right), \tag{15}$$

where $\mathbf{X}$ and $\mathbf{Z}$ are the corresponding image patches and $\|\cdot\|_F$ is the Frobenius norm for matrices.

## 4 Finding rank deficient reduced sets

In our approach we consider a special class of the approximations given by (3), namely those where the RSVs can be evaluated efficiently via separable correlations. In order to obtain such approximations, we use a constrained version of Burges' method. In particular, we restrict the RSV search space to the manifold spanned by all image patches that — viewed as matrices — have a fixed, small rank $r$ (which is to be chosen a priori by the user). To this end, the $\mathbf{Z}_i$ in equation (3) are replaced by their singular value decompositions

$$\mathbf{Z}_i \leftarrow \mathbf{U}_i \mathbf{S}_i \mathbf{V}_i^\top. \tag{16}$$

The rank constraint can then be imposed by allowing only the first $r$ diagonal elements of $\mathbf{S}_i$ to be non-zero. Note that this boils down to using an approximation of the form

$$\Psi'_r = \sum_{i=1}^{m'} \beta_i k(\mathbf{U}_{i,r} \mathbf{S}_{i,r} \mathbf{V}_{i,r}^\top, \cdot) \tag{17}$$

with $\mathbf{S}_{i,r}$ being $r \times r$ (diagonal) and $\mathbf{U}_{i,r}$, $\mathbf{V}_{i,r}$ being $h \times r$, $w \times r$ (orthogonal[2]) matrices, respectively. Analogously to (4) we fix $m'$ and $r$ and find $\mathbf{S}_{i,r}$, $\mathbf{U}_{i,r}$, $\mathbf{V}_{i,r}$ and $\beta_i$ that minimize the approximation error $\rho = \|\Psi - \Psi'_r\|_{\text{RKHS}}^2$. The minimization problem is solved via

gradient decent. Note that when computing gradients, the image-matrix notation (together with (14) or (15), and the equality $\|\mathbf{X}\|_F^2 = \mathrm{tr}(\mathbf{X}\mathbf{X}^\top)$) allows a straightforward computation of the kernel derivatives w.r.t. the components of the decomposed RSV image patches, i.e. the row, column and scale information in $\mathbf{V}_{i,r}$, $\mathbf{U}_{i,r}$ and $\mathbf{S}_{i,r}$, respectively. However, while the update rules for $\beta_i$ and $\mathbf{S}_{i,r}$ follow immediately from the respective derivatives, care must be taken in order to keep $\mathbf{U}_{i,r}$ and $\mathbf{V}_{i,r}$ orthogonal during optimization. This can be achieved through re-orthogonalization of these matrices after each gradient step.

In our current implementation, however, we perform those updates subject to the so-called Stiefel constraints [11]. Intuitively, this amounts to rotating (rather than translating) the columns of $\mathbf{U}_{i,r}$ and $\mathbf{V}_{i,r}$, which ensures that the resulting matrices are still orthogonal, i.e. lie on the Stiefel manifold. Let $\mathbb{S}(h,r)$ be the manifold of orthogonal $h \times r$ matrices, the $(h,r)$-Stiefel manifold. Further, let $\mathbf{U}_{i,r}^\perp$ denote an orthogonal basis for the orthogonal complement of the subspace spanned by the columns of $\mathbf{U}_{i,r}$. Now, given the 'free' gradient $\mathbf{G} = \partial\rho/\partial\mathbf{U}_{i,r}$ we compute the 'constrained' gradient

$$\hat{\mathbf{G}} = \mathbf{G} - \mathbf{U}_{i,r}\mathbf{G}^\top\mathbf{U}_{i,r}, \tag{18}$$

which is the projection of $\mathbf{G}$ onto the tangent space of $\mathbb{S}(h,r)$ at $\mathbf{U}_{i,r}$. The desired rotation is then given [11] by the (matrix) exponential of the $h \times h$ skew-symmetric matrix

$$\mathbf{A} = t \cdot \begin{pmatrix} \hat{\mathbf{G}}^\top\mathbf{U}_{i,r} & -(\hat{\mathbf{G}}^\top\mathbf{U}_{i,r}^\perp)^\top \\ \hat{\mathbf{G}}^\top\mathbf{U}_{i,r}^\perp & \mathbf{0} \end{pmatrix} \tag{19}$$

where $t$ is a user-defined step size parameter. For details, see [11]. A Matlab library is available at [12].

## 5 Experiments

This section shows the results of two experiments. The first part illustrates the behavior of rank deficient approximations for a face detection SVM in terms of the convergence rate and classification accuracy for different values of $r$. In the second part, we show how an actual face detection system, similar to that presented in [5], can be sped up using rank deficient RSVs. In both experiments we used the same training and validation set. It consisted of $19 \times 19$ gray level image patches containing 16081 manually collected faces (3194 of them kindly provided by Sami Romdhani) and 42972 non-faces automatically collected from a set of 206 background scenes. Each patch was normalized to zero mean and unit variance. The set was split into a training set (13331 faces and 35827 non-faces) and a validation set (2687 faces and 7145 non-faces). We trained a 1-norm soft margin SVM on the training set using a Gaussian kernel with $\sigma = 10$. The regularization constant $C$ was set to 1. The resulting decision function (1) achieved a hit rate of 97.3% at 1.0% false positives on the validation set using $m = 6910$ SVs. This solution served as the approximation target $\mathbf{\Psi}$ (see equation (2)) during the experiments described below.

### 5.1 Rank deficient faces

In order to see how $m'$ and $r$ affect the accuracy of our approximations, we compute rank deficient reduced sets for $m' = 1 \ldots 32$ and $r = 1 \ldots 3$ (the left array in Figure 1 illustrates the actual appearance of rank deficient RSVs for the $m' = 6$ case). Accuracy of the resulting decision functions is measured in ROC score (the area under the ROC curve) on the validation set. For the full SVM, this amounts to 0.99. The results for our approximations are depicted in Figure 2. As expected, we need a larger number of rank deficient RSVs than unconstrained RSVs to obtain similar classification accuracies, especially for small $r$. Nevertheless, the experiment points out two advantages of our method. First, a rank as

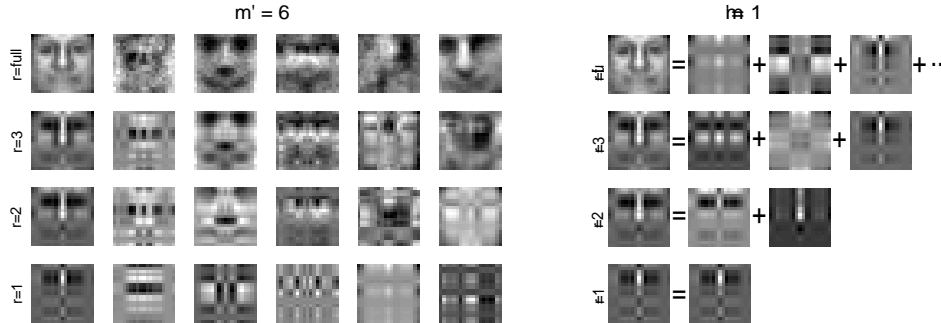

Figure 1: Rank deficient faces. The left array shows the RSVs ($\mathbf{Z}_i$) of the unconstrained (top row) and constrained ($r$ decreases from 3 to 1 down the remaining rows) approximations for $m' = 6$. Interestingly, the $r = 3$ RSVs are already able to capture face-like structures. This supports the fact that the classification accuracy for $r = 3$ is similar to that of the unconstrained approximations (cf. Figure 2, left plot). The right array shows the $m' = 1$ RSVs ($r = $ full, 3, 2, 1, top to bottom row) and their decomposition into rank one matrices according to (10). For the unconstrained RSV (first row) it shows an approximate (truncated) expansion based on the three leading singular vectors. While for $r = 3$ the decomposition is indeed similar to the truncated SVD, note how this similarity decreases for $r = 2, 1$. This illustrates that the approach is clearly different from simply finding unconstrained RSVs and *then* imposing the rank constraint via SVD (in fact, the norm (4) is smaller for the $r = 1$ RSV than for the leading singular vector of the $r = $ full RSV).

low as three seems already sufficient for our face detection SVM in the sense that for equal sizes $m'$ there is no significant loss in accuracy compared to the unconstrained approximation (at least for $m' > 2$). The associated speed benefit over unconstrained RSVs is shown in the right plot of Figure 2: the rank three approximations achieve accuracies similar to the unconstrained functions, while the number of operations reduces to less than a third. Second, while for unconstrained RSVs there is no solution with a number of operations smaller than $h \cdot w = 361$ (in the right plot, this is the region beyond the left end of the solid line), there exist rank deficient functions which are not only much faster than this, but yield considerably higher accuracies. This property will be exploited in the next experiment.

## 5.2   A cascade-based face detection system

In this experiment we built a cascade-based face detection system similar to [5, 6], i.e. a cascade of RSV approximations of increasing size $m'$. As the benefit of a cascaded classifier heavily depends on the speed of the first classifier which has to be evaluated on the whole image [5, 6], our system uses a rank deficient approximation as the first stage. Based on the previous experiment, we chose the $m' = 3$, $r = 1$ classifier. Note that this function yields an ROC score of 0.9 using 114 multiply-adds, whereas the simplest possible unconstrained approximation $m' = 1$, $r = $ full needs 361 multiply-adds to achieve a ROC score of only 0.83 (cf. Figure 2). In particular, if the threshold of the first stage is set to yield a hit rate of 95% on the validation set, scanning the MIT+CMU set (130 images, 507 faces) with $m' = 3$, $r = 1$ discards 91.5% of the false positives, whereas the $m' = 1$, $r = $ full can only reject 70.2%. At the same time, when scanning a $320 \times 240$ image[3], the three separable convolutions plus nonlinearity require 55ms, whereas the single, full kernel evaluation takes 208ms on a Pentium 4 with 2.8 GHz. Moreover, for the unconstrained

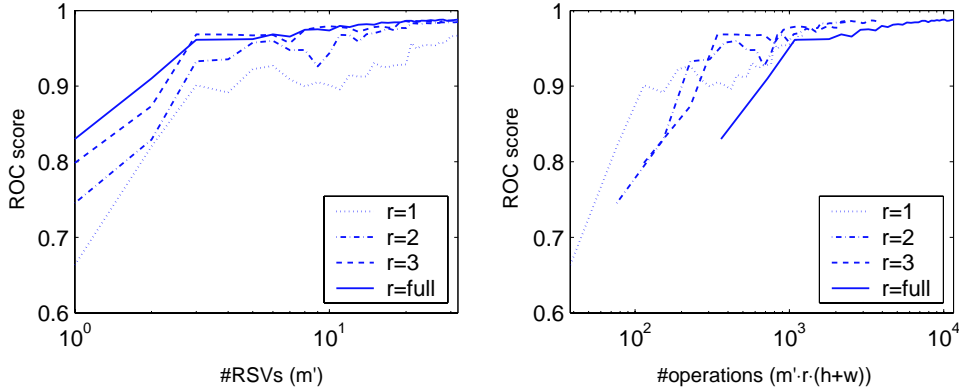

Figure 2: Effect of the rank parameter $r$ on classification accuracies. The left plots shows the ROC score of the rank deficient RSV approximations (cf. Section 4) for varying set sizes ($m' = 1 \ldots 32$, on a logarithmic scale) and ranks ($r = 1 \ldots 3$). Additionally, the solid line shows the accuracy of the RSVs without rank constraint (cf. Section 2), here denoted by $r = $ full. The right plot shows the same four curves, but plotted against the number of operations needed for the evaluation of the corresponding decision function when scanning large images (i.e. $m' \cdot r \cdot (h + w)$ with $h = w = 19$), also on a logarithmic scale.

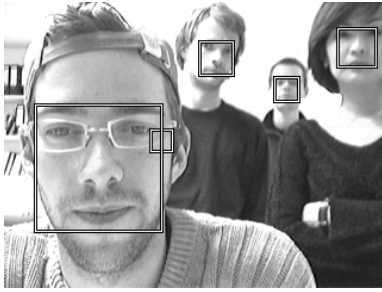

Figure 3: A sample output from our demonstration system (running at 14 frames per second). In this implementation, we reduced the number of false positives by adjusting the threshold of the final classifier. Although this reduces the number of detections as well, the results are still satisfactory. This is probably due to the fact that the MIT+CMU set contains several images of very low quality that are not likely to occur in our setting, using a good USB camera.

cascade to catch up in terms of accuracy, the (at least) $m' = 2$, $r = $ full classifier (also with an ROC score of roughly 0.9) should be applied afterwards, requiring another $0.3 * 2 * 208$ ms $\approx 125$ms.

The subsequent stages of our system consist of unconstrained RSV approximations of size $m' = 4, 8, 16, 32$, respectively. These sizes were chosen such that the number of false positives roughly halves after each stage, while the number of correct detections remains close to 95% on the validation set (with the decision thresholds adjusted accordingly). To eliminate redundant detections, we combine overlapping detections via averaging of position and size if they are closer than 0.15 times the estimated patch size. This system yields 93.1% correct detections and 0.034% false positives on the MIT+CMU set. The current system was incorporated into a demo application (Figure 3). For optimal performance, we re-compiled our system using the Intel compiler (ICC). The application now classifies a 320x240 image within 54ms (vs. 238ms with full rank RSVs only) on a 2.8 GHz PC. To further reduce the number of false positives, additional bootstrapped (as in [5]) stages need to be added to the cascade. Note that this will not significantly affect the speed of our system (currently 14 frames per second) since 0.034% false positives amounts to merely 47 patches to be processed by subsequent classifiers.

# 6 Discussion

We have presented a new reduced set method for SVMs in image processing, which creates sparse kernel expansions that can be evaluated via separable filters. To this end, the user-defined rank (the number of separable filters into which the RSVs are decomposed) provides a mechanism to control the tradeoff between accuracy and speed of the resulting approximation. Our experiments show that for face detection, the use of rank deficient RSVs leads to a significant speedup without losing accuracy. Especially when rough approximations are required, our method gives superior results compared to the existing reduced set methods since it allows for a finer granularity which is vital in cascade-based detection systems. Another property of our approach is simplicity. At run-time, rank deficient RSVs can be used together with unconstrained RSVs or SVs using the same canonical image representation. As a result, the required changes in existing code, such as in [5], are small. In addition, our approach allows the use of off-the-shelf image processing libraries for separable convolutions. Since such operations are essential in image processing, there exist many (often highly optimized) implementations. Finally, the method can well be used to train a neural network, i.e. to go directly from the training data to a sparse, separable function as opposed to taking the SVM 'detour'. A comparison of that approach to the present one, however, remains to be done.

## Footnotes

[1] In other words, the flatter the spectrum of $\mathbf{H}\mathbf{H}^\top$, the less benefit can be expected from (11).

[2]In this paper we call a non-square matrix orthogonal if its columns are pairwise orthogonal and have unit length.

[3]For multi-scale processing the detectors are evaluated on an image pyramid with 12 different scales using a scale decay of 0.75. This amounts to scanning 140158 patches for a $320 \times 240$ image.

# References

[1] E. Osuna and F. Girosi. Reducing the run-time complexity in support vector machines. In B. Schölkopf, C. J. C. Burges, and A. J. Smola, editors, *Advances in Kernel Methods — Support Vector Learning*, pages 271–284, Cambridge, MA, 1999. MIT Press.

[2] C. J. C. Burges. Simplified support vector decision rules. In *International Conference on Machine Learning*, pages 71–77, 1996.

[3] C. J. C. Burges and B. Schölkopf. Improving the accuracy and speed of support vector machines. In M. C. Mozer, M. I. Jordan, and T. Petsche, editors, *Advances in Neural Information Processing Systems*, volume 9, page 375. MIT Press, 1997.

[4] E. Osuna, R. Freund, and F. Girosi. Training support vector machines: an application to face detection. In *Proceedings IEEE Conference on Computer Vision and Pattern Recognition*, 1997.

[5] S. Romdhani, P. Torr, B.Schölkopf, and A. Blake. Computationally efficient face detection. In *Proceedings of the International Conference on Computer Vision*, pages 695–700, 2001.

[6] P. Viola and M. Jones. Rapid object detection using a boosted cascade of simple features. In *Proceedings IEEE Conference on Computer Vision and Pattern Recognition*, 2001.

[7] G. Blanchard and D. Geman. Hierarchical testing designs for pattern recognition. Technical Report 2003-07, Universit Paris-Sud, 2003.

[8] B. Heisele, T. Poggio, and M. Pontil. Face detection in still gray images. AI Memo 1687, MIT, May 2000. CBCL Memo 187.

[9] S. Ben-Yacoub, B. Fasel, and J. Luettin. Fast face detection using MLP and FFT. In *Proceedings International Conference on Audio and Video-based Biometric Person Authentication*, 1999.

[10] H. A. Rowley, S. Baluja, and T. Kanade. Neural network-based face detection. *IEEE Transactions on Pattern Analysis and Machine Intelligence*, 20(1):23–38, January 1998.

[11] A. Edelman, T. Arias, and S. Smith. The geometry of algorithms with orthogonality constraints. *SIAM Journal on Matrix Analysis and Applications*, 20:303–353, 1998.

[12] RDRSLIB – a matlab library for rank deficient reduced sets in object detection, http://www.kyb.mpg.de/bs/people/kienzle/rdrs/rdrs.htm.
